# The Epoch-Greedy Algorithm for Contextual Multi-armed Bandits

**John Langford**
Yahoo! Research
jl@yahoo-inc.com

**Tong Zhang**
Department of Statistics
Rutgers University
tongz@rci.rutgers.edu

## Abstract

We present Epoch-Greedy, an algorithm for contextual multi-armed bandits (also known as bandits with side information). Epoch-Greedy has the following properties:

1. No knowledge of a time horizon $T$ is necessary.
2. The regret incurred by Epoch-Greedy is controlled by a sample complexity bound for a hypothesis class.
3. The regret scales as $O(T^{2/3}S^{1/3})$ or better (sometimes, much better). Here $S$ is the complexity term in a sample complexity bound for standard supervised learning.

## 1 Introduction

The standard $k$-armed bandits problem has been well-studied in the literature (Lai & Robbins, 1985; Auer et al., 2002; Even-dar et al., 2006, for example). It can be regarded as a repeated game between two players, with every stage consisting of the following: The world chooses $k$ rewards $r_1, ..., r_k \in [0,1]$; the player chooses an arm $i \in \{1, k\}$ without knowledge of the world's chosen rewards, and then observes the reward $r_i$. The contextual bandits setting considered in this paper is the same except for a modification of the first step, in which the player also observes context information $x$ which can be used to determine which arm to pull.

The contextual bandits problem has many applications and is often more suitable than the standard bandits problem, because settings with *no* context information are rare in practice. The setting considered in this paper is directly motivated by the problem of matching ads to web-page contents on the internet. In this problem, a number of ads (arms) are available to be placed on a number of web-pages (context information). Each page visit can be regarded as a random draw of the context information (one may also include the visitor's online profile as context information if available) from an underlying distribution that is not controlled by the player. A certain amount of revenue is generated when the visitor clicks on an ad. The goal is to put the most relevant ad on each page to maximize the expected revenue. Although one may potentially put multiple ads on each web-page, we focus on the problem that only one ad is placed on each page (which is like pulling an arm given context information). The more precise definition is given in Section 2.

**Prior Work**. The problem of bandits with context has been analyzed previously (Pandey et al., 2007; Wang et al., 2005), typically under additional assumptions such as a correct prior or knowledge of the relationship between the arms. This problem is also known as associative reinforcement learning (Strehl et al., 2006, for example) or bandits with side information. A few results under as weak or weaker assumptions are directly comparable.

1. The Exp4 algorithm (Auer et al., 1995) notably makes *no* assumptions about the world. Epoch-Greedy has a worse regret bound in $T$ ($O(T^{2/3})$ rather than $O(T^{1/2})$) and is only

analyzed under an IID assumption. An important advantage of Epoch-Greedy is a much better dependence on the size of the set of predictors. In the situation where the number of predictors is infinite but with finite VC-Dimension $d$, Exp4 has a vacuous regret bound while Epoch-Greedy has a regret bound no worse than $O(T^{2/3}(\ln m)^{1/3})$. Sometimes we can achieve much better dependence on $T$, depending on the structure of the hypothesis space. For example, we will show that it is possible to achieve $O(\ln T)$ regret bound using Epoch-Greedy, while this is not possible with Exp4 or any simple modification of it. Another substantial advantage is reduced computational complexity. The ERM step in Epoch-Greedy can be replaced with any standard learning algorithm that achieves approximate loss minimization, making guarantees that degrade gracefully with the approximation factor. Exp4 on the other hand requires computation proportional to the explicit count of hypotheses in a hypothesis space.

2. The random trajectories method (Kearns et al., 2000) for learning policies in reinforcement learning with hard horizon $T = 1$ is essentially the same setting. In this paper, bounds are stated for a batch oriented setting where examples are formed and then used for choosing a hypothesis. Epoch-Greedy takes advantage of this idea, but it also has analysis which states that it trades off the number of exploration and exploitation steps so as to maximize the sum of rewards incurred during both exploration and exploitation.

**What we do**. We present and analyze the Epoch-Greedy algorithm for multiarmed bandits with context. This has all the nice properties stated in the abstract, resulting in a practical algorithm for solving this problem.

The paper is broken up into the following sections.

1. In Section 2 we present basic definitions and background.
2. Section 3 presents the Epoch-Greedy algorithm along with a regret bound analysis which holds without knowledge of $T$.
3. Section 4 analyzes the instantiation of the Epoch-Greedy algorithm in several settings.

## 2 Contextual bandits

We first formally define contextual bandit problems and algorithms to solve them.

**Definition 2.1 (Contextual bandit problem)** *In a contextual bandits problem, there is a distribution $P$ over $(x, r_1, ..., r_k)$, where $x$ is context, $a \in \{1, \ldots, k\}$ is one of the $k$ arms to be pulled, and $r_a \in [0, 1]$ is the reward for arm $a$. The problem is a repeated game: on each round, a sample $(x, r_1, ..., r_k)$ is drawn from $P$, the context $x$ is announced, and then for precisely one arm $a$ chosen by the player, its reward $r_a$ is revealed.*

**Definition 2.2 (Contextual bandit algorithm)** *A contextual bandits algorithm $\mathcal{B}$ determines an arm $a \in \{1, \ldots, k\}$ to pull at each time step $t$, based on the previous observation sequence $(x_1, a_1, r_{a,1}), \ldots, (x_{t-1}, a_{t-1}, r_{a,t-1})$, and the current context $x_t$.*

Our goal is to maximize the expected total reward $\sum_{t=1}^{T} \mathbf{E}_{(x_t, \vec{r}_t) \sim P} [r_{a,t}]$. Note that we use the notation $r_{a,t} = r_{a_t}$ to improve readability. Similar to supervised learning, we assume that we are given a set $\mathcal{H}$ consisting of hypotheses $h : \mathcal{X} \rightarrow \{1, \ldots, k\}$. Each hypothesis maps side information $x$ to an arm $a$. A natural goal is to choose arms to compete with the best hypothesis in $\mathcal{H}$. We introduce the following definition.

**Definition 2.3 (Regret)** *The expected reward of a hypothesis $h$ is*

$$R(h) = \mathbf{E}_{(x, \vec{r}) \sim D} \left[ r_{h(x)} \right].$$

*Consider any contextual bandits algorithm $\mathcal{B}$. Let $Z^T = \{(x_1, \vec{r}_1), \ldots, (x_T, \vec{r}_T)\}$, and the expected regret of $\mathcal{B}$ with respect to a hypothesis $h$ be:*

$$\Delta R(\mathcal{B}, h, T) = T R(h) - \mathbf{E}_{Z^T \sim P^T} \sum_{t=1}^{T} r_{\mathcal{B}(x), t}.$$

*The expected regret of $\mathcal{B}$ up to time $T$ with respect to hypothesis space $\mathcal{H}$ is defined as*

$$\Delta R(\mathcal{B}, \mathcal{H}, T) = \sup_{h \in \mathcal{H}} \Delta R(\mathcal{B}, h, T).$$

The main challenge of the contextual bandits problem is that when we pull an arm, rewards of other arms are not observed. Therefore it is necessary to try all arms (explore) in order to form an accurate estimation. In this context, methods we investigate in the paper make explicit distinctions between *exploration* and *exploitation* steps. In an exploration step, the goal is to form unbiased samples by randomly pulling all arms to improve the accuracy of learning. Because it does not focus on the best arm, this step leads to large immediate regret but can potentially reduce regret for the future exploitation steps. In an exploitation step, the learning algorithm suggests the best hypothesis learned from the samples formed in the exploration steps, and the arm given by the hypothesis is pulled: the goal is to maximize immediate reward (or minimize immediate regret). Since the samples in the exploitation steps are biased (toward the arm suggested by the learning algorithm using previous exploration samples), we do not use them to learn the hypothesis for the future steps. That is, in methods we consider, exploitation does not help us to improve learning accuracy for the future.

More specifically, in an exploration step, in order to form unbiased samples, we pull an arm $a \in \{1, \ldots, k\}$ uniformly at random. Therefore the expected regret comparing to the best hypothesis in $\mathcal{H}$ can be as large as $O(1)$. In an exploitation step, the expected regret can be much smaller. Therefore a central theme we examine in this paper is to balance the trade-off between exploration and exploitation, so as to achieve a small overall expected regret up to some time horizon $T$.

Note that if we decide to pull a specific arm $a$ with side information $x$, we do not observe rewards $r_{a'}$ for $a' \neq a$. In order to apply standard sample complexity analysis, we first show that exploration samples, where $a$ is picked uniformly at random, can create a standard learning problem without missing observations. This is simply achieved by setting fully observed rewards $r'$ such that

$$r'_{a'}(r_a) = kI(a' = a)r_a, \tag{1}$$

where $I(\cdot)$ is the indicator function. The basic idea behind this transformation from partially observed to fully observed data dates back to the analysis of "Sample Selection Bias" (Heckman, 1979). The above rule is easily generalized to other distribution over actions $p(a)$ by replacing $k$ with $1/p(a)$.

The following lemma shows that this method of filling missing reward components is unbiased.

**Lemma 2.1** *For all arms $a'$: $\mathbf{E}_{\vec{r} \sim P|x} [r_{a'}] = \mathbf{E}_{\vec{r} \sim P|x, a \sim U(1,\ldots,k)} [r'_{a'}(r_a)]$. Therefore for any hypothesis $h(x)$, we have $R(h) = \mathbf{E}_{(x,\vec{r}) \sim P, a \sim U(1,\ldots,k)} \left[ r'_{h(x)}(r_a) \right]$.*

**Proof** We have:

$$\mathbf{E}_{\vec{r} \sim P|x, a \sim U(1,\ldots,k)} [r'_{a'}(r_a)] = \mathbf{E}_{\vec{r} \sim P|x} \sum_{a=1}^{k} k^{-1} [r'_{a'}(r_a)]$$

$$= \mathbf{E}_{\vec{r} \sim P|x} \sum_{a=1}^{k} k^{-1} [kr_a I(a' = a)] = \mathbf{E}_{\vec{r} \sim P|x} [r_{a'}].$$

∎

Lemma 2.1 implies that we can estimate reward $R(h)$ of any hypothesis $h(x)$ using expectation with respect to exploration samples $(x, a, r_a)$. The right hand side can then be replaced by empirical samples as $\sum_t I(h(x_t) = a_t) r_{a,t}$ for hypotheses in a hypothesis space $\mathcal{H}$. The quality of this estimation can be obtained with uniform convergence learning bounds.

## 3 Exploration with the Epoch-Greedy algorithm

The problem of treating contextual bandits as standard bandits is that the information in $x$ is lost. That is, the optimal arm to pull should be a function of the context $x$, but this is not captured by the

standard bandits setting. An alternative approach is to regard each hypothesis $h$ as a separate artificial "arm", and then apply a standard bandits algorithm to these artificial arms. Using this approach, let $m$ be the number of hypotheses, we can get a bound of $O(m)$. However, this solution ignores the fact that many hypotheses can share the same arm so that choosing an arm yields information for many hypotheses. For this reason, with a simple algorithm, we can get a bound that depends on $m$ logarithmically, instead of $O(m)$ as would be the case for the standard bandits solution discussed above.

As discussed earlier, the key issue in the algorithm is to determine when to explore and when to exploit, so as to achieve appropriate balance. If we are given the time horizon $T$ in advance, and would like to optimize performance with the given $T$, then it is always advantageous to perform a first phase of exploration steps, followed by a second phase of exploitation steps (until time step $T$). The reason that there is no advantage to take any exploitation step before the last exploration step is: by switching the two steps, we can more accurately pick the optimal hypothesis in the exploitation step due to more samples from exploration. With fixed $T$, assume that we have taken $n$ steps of exploration, and obtain an average regret bound of $\epsilon_n$ for each exploitation step at the point, then we can bound the regret of the exploration phase as $n$, and the exploitation phase as $\epsilon_n(T - n)$. The total regret is $n + (T - n)\epsilon_n$. Using this bound, we shall switch from exploration to exploitation at the point $n$ that minimizes the sum.

Without knowing $T$ in advance, but with the same generalization bound, we can run exploration/exploitation in epochs, where at the beginning of each epoch $\ell$, we perform one step of exploration, followed by $\lceil 1/\epsilon_n \rceil$ steps of exploitation. We then start the next epoch. After epoch $L$, the total average regret is no more than $\sum_{n=1}^{L}(1 + \epsilon_n\lceil 1/\epsilon_n \rceil) \leq 3L$. Moreover, the epoch $L_*$ containing $T$ is no more than the optimal regret bound $\min_n[n + (T - n)\epsilon_n]$ (with known $T$ and optimal stopping point). Therefore the performance of our method (which does not need to know $T$) is no worse than three time the optimal bound with known $T$ and optimal stopping point. This motivates a modified algorithm in Figure 1. The idea described above is related to forcing in (Lai & Yakowitz, 1995).

**Proposition 3.1** *Consider a sequence of nonnegative and monotone non-increasing numbers $\{\epsilon_n\}$. Let $L_* = \min\{L : \sum_{\ell=1}^{L}(1 + \lceil 1/\epsilon_\ell \rceil) \geq T\}$, then*

$$L_* \leq \min_{n \in [0,T]}[n + (T - n)\epsilon_n].$$

**Proof** Let $n_* = \arg\min_{n \in [0,T]}[n + (T - n)\epsilon_n]$. The bound is trivial if $n_* \geq L_*$. We only need consider the case $n_* \leq L_* - 1$. By assumption, $\sum_{\ell=1}^{L_*-1}(1 + 1/\epsilon_\ell) \leq T - 1$. Since $\sum_{\ell=1}^{L_*-1} 1/\epsilon_\ell \geq \sum_{\ell=n_*}^{L_*-1} 1/\epsilon_\ell \geq (L_* - n_*)1/\epsilon_{n_*}$, we have $L_* - 1 + (L_* - n_*)1/\epsilon_{n_*} \leq T - 1$. Rearranging, we have $L_* \leq n_* + (T - L_*)\epsilon_{n_*}$. ∎

In Figure 1, $s(Z_1^n)$ is a sample-dependent (integer valued) exploitation step count. Proposition 3.1 suggests that choosing $s(Z_1^n) = \lceil 1/\epsilon_n(Z_1^n) \rceil$, where $\epsilon_n(Z_1^n)$ is a sample dependent average generalization bound, yields performance comparable to the optimal bound with known time horizon $T$.

**Definition 3.1 (Epoch-Greedy Exploitation Cost)** *Consider a hypothesis space $\mathcal{H}$ consisting of hypotheses that take values in $\{1, 2, \ldots, k\}$. Let $Z_t = (x_t, a_t, r_{a,t})$ for $i = 1, \ldots, n$ be independent random samples, where $a_i$ is uniform randomly distributed in $\{1, \ldots, k\}$, and $r_{a,t} \in [0, 1]$ is the observed (random) reward. Let $Z_1^n = \{Z_1, \ldots, Z_n\}$, and the empirical reward maximization estimator*

$$\hat{h}(Z_1^n) = \arg\max_{h \in \mathcal{H}} \sum_{t=1}^{n} r_{a,t} I(h(x_t) = a_t).$$

*Given any fixed $n$, $\delta \in [0, 1]$, and observation $Z_1^n$, we denote by $s(Z_1^n)$ a data-dependent exploitation step count. Then the per-epoch exploitation cost is defined as:*

$$\mu_n(\mathcal{H}, s) = \mathbf{E}_{Z_1^n}\left(\sup_{h \in \mathcal{H}} R(h) - R(\hat{h}(Z_1^n))\right)s(Z_1^n).$$

Figure 1: Exploration by $\epsilon$-greedy in epochs

**Theorem 3.1** *For all $T, n_\ell, L$ such that: $T \le L + \sum_{\ell=1}^{L} n_\ell$, the expected regret of Epoch-Greedy in Figure 1 is bounded by*

$$\Delta R(\text{Epoch-Greedy}, \mathcal{H}, T) \le L + \sum_{\ell=1}^{L} \mu_\ell(\mathcal{H}, s) + T \sum_{\ell=1}^{L} P[s(Z_1^\ell) < n_\ell].$$

This theorem statement is very general, because we want to allow sample dependent bounds to be used. When sample-independent bounds are used the following simple corollary holds:

**Corollary 3.1** *Assume we choose $s(Z_1^\ell) = s_\ell \le \lfloor 1/\mu_\ell(\mathcal{H}, 1) \rfloor$ ($\ell = 1, \ldots$), and let $L_T = \arg\min_L \{L : L + \sum_{\ell=1}^{L} s_\ell \ge T\}$. Then the expected regret of Epoch-Greedy in Figure 1 is bounded by*

$$\Delta R(\text{Epoch-Greedy}, \mathcal{H}, T) \le 2L_T.$$

**Proof** (of the main theorem) Let $\mathcal{B}$ be the Epoch-Greedy algorithm. One of the following events will occur:

- A: $s(Z_1^\ell) < n_\ell$ for some $\ell = 1, \ldots, L$.
- B: $s(Z_1^\ell) \ge n_\ell$ for all $\ell = 1, \ldots, L$.

If event A occurs, then since each reward is in [0,1], up to time $T$, regret cannot be larger than $T$. Thus the total expected contribution of A to the regret $\Delta R(\mathcal{B}, \mathcal{H}, T)$ is at most

$$TP(A) \le T \sum_{\ell=1}^{L} P[s(Z_1^\ell) < n_\ell]. \tag{2}$$

If event B occurs, then $t_{\ell+1} - t_\ell \ge n_\ell + 1$ for $\ell = 1, \ldots, L$, and thus $t_{L+1} > T$. Therefore the expected contribution of B to the regret $\Delta R(\mathcal{B}, \mathcal{H}, T)$ is at most the sum of expected regret in the first $L$ epochs.

By definition and construction, after the first step of epoch $\ell$, $W_\ell$ consists of $\ell$ random observations $Z_j = (x_j, a_j, r_{a,j})$ where $a_j$ is drawn uniformly at random from $\{1, \ldots, k\}$, and $j = 1, \ldots, \ell$. This is independent of the number of exploitation steps before epoch $\ell$. Therefore we can treat $W_\ell$ as $\ell$ independent samples. This means that the expected regret associated with exploitation steps in epoch $\ell$ is $\mu_\ell(\mathcal{H}, s)$. Since the exploration step in each epoch contributes at most 1 to the

expected regret, the total expected regret for each epoch $\ell$ is at most $1 + \mu_\ell(\mathcal{H}, s)$. Therefore the total expected regret for epochs $\ell = 1, \ldots, L$ is at most $L + \sum_{\ell=1}^{L} \mu_\ell(\mathcal{H}, s)$. Combined with (2), we obtain the desired bound. ∎

In the theorem, we bound the expected regret of each exploration step by one. Clearly this assumes the worst case scenario and can often be improved. Some consequences of the theorem with specific function classes are given in Section 4.

# 4 Examples

Theorem 3.1 is quite general. In this section, we present a few simple examples to illustrate the potential applications.

## 4.1 Finite hypothesis space worst case bound

Consider the finite hypothesis space situation, with $m = |\mathcal{H}| < \infty$. We apply Theorem 3.1 with a worst-case deviation bound.

Let $x_1, \ldots, x_n \in [0, k]$ be iid random variables, such that $\mathbf{E}x_i \leq 1$, then Bernstein inequality implies that there exists a constant $c_0 > 0$ such that $\forall \eta \in (0, 1)$, with probability $1 - \eta$:

$$\left| \sum_{i=1}^{n} x_i - \sum_{i=1}^{n} \mathbf{E}x_i \right| \leq c_0 \sqrt{\ln(1/\eta) \sum_{i=1}^{n} \mathbf{E}x_i^2} + c_0 k \ln(1/\eta) \leq c_0 \sqrt{nk \ln(1/\eta)} + c_0 k \ln(1/\eta).$$

It follows that there exists a universal constant $c > 0$ such that

$$\mu_n(\mathcal{H}, 1) \leq c^{-1} \sqrt{k \ln m / n}.$$

Therefore in Figure 1, if we choose

$$s(Z_1^\ell) = \lfloor c\sqrt{\ell/(k \ln m)} \rfloor,$$

then $\mu_\ell(\mathcal{H}, s) \leq 1$: this is consistent with the choice recommended in Proposition 3.1.

In order to obtain a performance bound of this scheme using Theorem 3.1, we can simply take

$$n_\ell = \lfloor c\sqrt{\ell/(k \ln m)} \rfloor.$$

This implies that $P(s(Z_1^\ell) < n_\ell) = 0$. Moreover, with this choice, for any $T$, we can pick an $L$ that satisfies the condition $T \leq \sum_{\ell=1}^{L} n_\ell$. It implies that there exists a universal constant $c' > 0$ such that for any given $T$, we can take

$$L = \lfloor c'T^{2/3}(k \ln m)^{1/3} \rfloor$$

in Theorem 3.1.

In summary, if we choose $s(Z_1^\ell) = \lfloor c\sqrt{\ell/(k \ln m)} \rfloor$ in Figure 1, then

$$\Delta(\text{Epoch-Greedy}, \mathcal{H}, T) \leq 2L \leq 2c'T^{2/3}(k \ln m)^{1/3}.$$

Reducing the problem to standard bandits, as discussed at the beginning of Section 3, leads to a bound of $O(m \ln T)$ (Lai & Robbins, 1985; Auer et al., 2002). Therefore when $m$ is large, the Epoch-Greedy algorithm in Figure 1 can perform significantly better. In this particular situation, Epoch-Greedy does not do as well as Exp4 in (Auer et al., 1995), which implies a regret of $O(\sqrt{kT \ln m})$. However, the advantage of Epoch-Greedy is that any learning bound can be applied. For many hypothesis classes, the $\ln m$ factor can be improved for Epoch-Greedy. In fact, a similar result can be obtained for classes with infinitely many hypotheses but finite VC dimensions. Moreover, as we will see next, under additional assumptions, it is possible to obtain much better bounds in terms of $T$ for Epoch-Greedy, such as $O(k \ln m + k \ln T)$. This extends the classical $O(\ln T)$ bound for standard bandits, and is not possible to achieve using Exp4 or simple variations of it.

## 4.2 Finite hypothesis space with unknown expected reward gap

This example illustrates the importance of allowing sample-dependent $s(Z_1^\ell)$. We still assume a finite hypothesis space, with $m = |\mathcal{H}| < \infty$. However, we would like to improve the performance bound by imposing additional assumptions. In particular we note that the standard bandits problem has regret of the form $O(\ln T)$ while in the worst case, our method for the contextual bandits problem has regret $O(T^{2/3})$. A natural question is then: what are the assumptions we can impose so that the Epoch-Greedy algorithm can have a regret of the form $O(\ln T)$.

The main technical reason that the standard bandits problem has regret $O(\ln T)$ is that the expected reward of the best bandit and that of the second best bandit has a gap: the constant hidden in the $O(\ln T)$ bound depends on this gap, and the bound becomes trivial (infinity) when the gap approaches zero. In this example we show that a similar assumption for contextual bandits problems leads to a similar regret bound of $O(\ln T)$ for the Epoch-Greedy algorithm.

Let $\mathcal{H} = \{h_1, \ldots, h_m\}$, and assume without loss of generality that $R(h_1) \geq R(h_2) \geq \cdots \geq R(h_m)$. Suppose that we know that $R(h_1) \geq R(h_2) + \Delta$ for some $\Delta > 0$, but the value of $\Delta$ is not known in advance.

Although $\Delta$ is not known, it can be estimated from the data $Z_1^n$. Let the empirical reward of $h \in \mathcal{H}$ be

$$\hat{R}(h|Z_1^n) = \frac{k}{n} \sum_{t=1}^n r_{a,t} I(h(x_t) = a_t).$$

Let $\hat{h}_1$ be the hypothesis with highest empirical reward on $Z_1^n$, and $\hat{h}_2$ be the hypothesis with second highest empirical reward. We define the empirical gap as

$$\hat{\Delta}(Z_1^n) = \hat{R}(\hat{h}_1|Z_1^n) - \hat{R}(\hat{h}_2|Z_1^n).$$

Let $h_1$ be the hypothesis with the highest true expected reward, then we suffer a regret when $\hat{h}_1 \neq h_1$. Again, the standard large deviation bound implies that there exists a universal constant $c > 0$ such that for all $j \geq 1$:

$$P(\hat{\Delta}(Z_1^n) \geq (j-1)\Delta, \hat{h}_1 \neq h_1) \leq me^{-ck^{-1}n(1+j^2)\Delta^2}$$
$$P(\hat{\Delta}(Z_1^n) \leq 0.5\Delta) \leq me^{-ck^{-1}n\Delta^2}.$$

Now we can set $s(Z_1^n) = \lfloor m^{-1}e^{(2k)^{-1}cn\hat{\Delta}(Z_1^n)^2} \rfloor$. With this choice, there exists a constant $c' > 0$ such that

$$\mu_n(\mathcal{H}, s) \leq \sum_{j=1}^{\lceil \Delta^{-1} \rceil} \sup\{s(Z_1^n) : \hat{\Delta}(Z_1^n) \leq j\Delta\} P(\hat{\Delta}(Z_1^n) \in [(j-1)\Delta, j\Delta], \hat{h}_1 \neq h_1)$$

$$\leq \sum_{j=1}^{\lceil \Delta^{-1} \rceil} m^{-1}e^{(2k)^{-1}cnj^2\Delta^2} P(\hat{\Delta}(Z_1^n) \in [(j-1)\Delta, j\Delta], \hat{h}_1 \neq h_1)$$

$$\leq \sum_{j=1}^{\lceil \Delta^{-1} \rceil} e^{(2k)^{-1}cnj^2\Delta^2 - ck^{-1}n(1+j^2)\Delta^2}$$

$$\leq \sum_{j=1}^{\lceil \Delta^{-1} \rceil} e^{-ck^{-1}n(0.5j^2+1)\Delta^2}$$

$$\leq c'\sqrt{k/n}\Delta^{-1}e^{-ck^{-1}n\Delta^2}.$$

There exists a constant $c'' > 0$ such that for any $L$:

$$\sum_{\ell=1}^L \mu_\ell(\mathcal{H}, s) \leq L + c' \sum_{\ell=1}^\infty \sqrt{k/\ell}\Delta^{-1}e^{-ck^{-1}\ell\Delta^2}$$

$$\leq L + c''k\Delta^{-2}.$$

Now, consider any time horizon $T$. If we set $n_\ell = 0$ when $\ell < L$, $n_L = T$, and

$$L = \left\lceil \frac{8k(\ln m + \ln(T+1))}{c\Delta^2} \right\rceil,$$

then

$$P(s(Z_1^L) \leq n_L) \leq P(\hat{\Delta}(Z_1^L) \leq 0.5\Delta) \leq m e^{-ck^{-1}L\Delta^2} \leq 1/T.$$

That is, if we choose $s(Z_1^n) = \lfloor m^{-1} e^{(2k)^{-1}cn\hat{\Delta}(Z_1^n)^2} \rfloor$ in Figure 1, then

$$\Delta R(\text{Epoch-Greedy}, \mathcal{H}, T) \leq 2L + 1 + c''k\Delta^{-2} \leq 2\left\lceil \frac{8k(\ln m + \ln(T+1))}{c\Delta^2} \right\rceil + 1 + c''k\Delta^{-2}.$$

The regret for this choice is $O(\ln T)$, which is better than $O(T^{2/3})$ of Section 4.1. However, the constant depends on the gap $\Delta$ which can be small. It is possible to combine the two strategies (that is, use the $s(Z_1^n)$ choice of Section 4.1 when $\hat{\Delta}(Z_1^n)$ is small) and obtain bounds that not only work well when the gap $\Delta$ is large, but also not much worse than the bound of Section 4.1 when $\Delta$ is small. As a special case, we can apply the method in this section to solve the standard bandits problem. The $O(k \ln T)$ bound of the Epoch-Greedy method matches those more specialized algorithms for the standard bandits problem, although our algorithm has a larger constant.

## 5  Conclusion

We consider a generalization of the multi-armed bandits problem, where observable context can be used to determine which arm to pull and investigate the sample complexity of the exploration/exploitation trade-off for the Epoch-Greedy algorithm.

The Epoch-Greedy algorithm analysis leaves one important open problem behind. Epoch-Greedy is much better at dealing with large hypothesis spaces or hypothesis spaces with special structures due to its ability to employ any data-dependent sample complexity bound. However, for finite hypothesis space, in the worst case scenario, Exp4 has better dependency on $T$. In such situations, it's possible that a better designed algorithm can achieve both strengths.

## References

Auer, P., Cesa-Bianchi, N., & Fischer, P. (2002). Finite time analysis of the multi-armed bandit problem. *Machine Learning*, *47*, 235–256.

Auer, P., Cesa-Bianchi, N., Freund, Y., & Schapire, R. E. (1995). Gambling in a rigged casino: The adversarial multi-armed bandit problem. *FOCS*.

Even-dar, E., Mannor, S., & Mansour, Y. (2006). Action elimination and stopping conditions for the multi-armed bandit and reinforcement learning problems. *JMLR*, *7*, 1079–1105.

Heckman, J. (1979). Sample selection bias as a specification error. *Econometrica*, *47*, 153–161.

Kearns, M., Mansour, Y., & Ng, A. Y. (2000). Approximate planning in large pomdps via reusable trajectories. *NIPS*.

Lai, T., & Robbins, H. (1985). Asymptotically efficient adaptive allocation rules. *Advances in Applied Mathematics*, *6*, 4–22.

Lai, T., & Yakowitz, S. (1995). Machine learning and nonparametric bandit theory. *IEEE TAC*, *40*, 1199–1209.

Pandey, S., Agarwal, D., Chakrabarti, D., & Josifovski, V. (2007). Bandits for taxonomies: a model-based approach. *SIAM Data Mining Conference*.

Strehl, A. L., Mesterharm, C., Littman, M. L., & Hirsh, H. (2006). Experience-efficient learning in associative bandit problems. *ICML*.

Wang, C.-C., Kulkarni, S. R., & Poor, H. V. (2005). Bandit problems with side observations. *IEEE Transactions on Automatic Control*, *50*, 338–355.

